# Timely Object Recognition

**Sergey Karayev**
UC Berkeley

**Tobias Baumgartner**
RWTH Aachen University

**Mario Fritz**
MPI for Informatics

**Trevor Darrell**
UC Berkeley

## Abstract

In a large visual multi-class detection framework, the timeliness of results can be crucial. Our method for *timely* multi-class detection aims to give the best possible performance at any single point after a start time; it is terminated at a deadline time. Toward this goal, we formulate a dynamic, closed-loop policy that infers the contents of the image in order to decide which detector to deploy next. In contrast to previous work, our method significantly diverges from the predominant greedy strategies, and is able to learn to take actions with deferred values. We evaluate our method with a novel *timeliness* measure, computed as the area under an Average Precision vs. Time curve. Experiments are conducted on the PASCAL VOC object detection dataset. If execution is stopped when only half the detectors have been run, our method obtains 66% better AP than a random ordering, and 14% better performance than an intelligent baseline. On the timeliness measure, our method obtains at least 11% better performance. Our method is easily extensible, as it treats detectors and classifiers as black boxes and learns from execution traces using reinforcement learning.

## 1  Introduction

In real-world applications of visual object recognition, performance is time-sensitive. In robotics, a small finite amount of processing power per unit time is all that is available for robust object detection, if the robot is to usefully interact with humans. In large-scale detection systems, such as image search, results need to be obtained quickly per image as the number of items to process is constantly growing. In such cases, an acceptable answer at a reasonable time may be more valuable than the best answer given too late.

A hypothetical system for vision-based advertising presents a case study: companies pay money to have their products detected in images on the internet. The system has different values (in terms of cost per click) and accuracies for different classes of objects, and the queue of unprocessed images varies in size. The detection strategy to maximize profit in such an environment has to exploit every inter-object context signal available to it, because there is not enough time to run detection for all classes.

What matters in the real world is timeliness, and either not all images can be processed or not all classes can be evaluated in a detection task. Yet the conventional approach to evaluating visual recognition does not consider efficiency, and evaluates performance independently across classes. We argue that the key to tackling problems of dynamic recognition resource allocation is to start asking a new question: *What is the best performance we can get on a budget?*

Taking the task of object detection, we propose a new *timeliness* measure of performance vs. time (shown in Figure 1). We present a method that treats different detectors and classifiers as black boxes, and uses reinforcement learning to learn a dynamic policy for selecting actions to achieve the highest performance under this evaluation.

Specifically, we run scene context and object class detectors over the whole image sequentially, using the results of detection obtained so far to select the next actions. Evaluating on the PASCAL

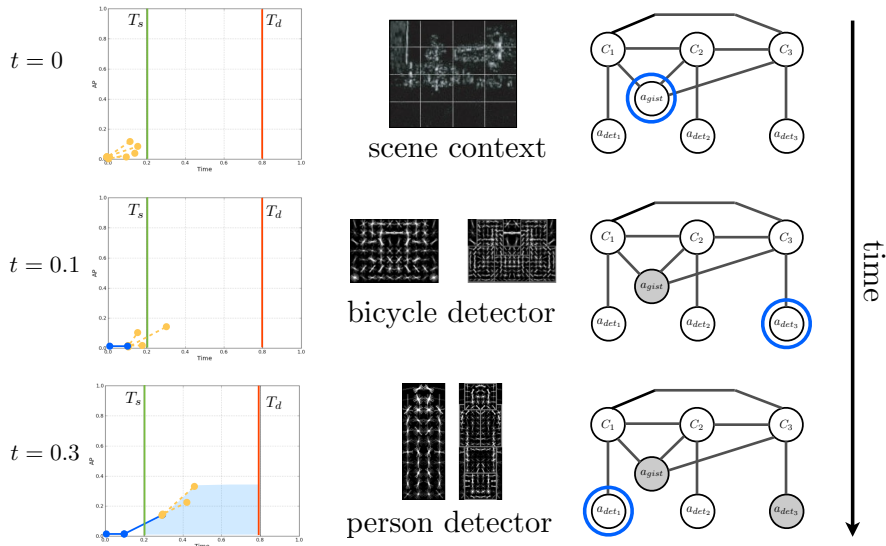

Figure 1: A sample trace of our method. At each time step beginning at $t = 0$, potential actions are considered according to their predicted *value*, and the maximizing action is picked. The selected action is performed and returns *observations*. Different actions return different observations: a detector returns a list of detections, while a scene context action simply returns its computed feature. The *belief model* of our system is updated with the observations, which influences the selection of the next action. The final evaluation of a detection episode is the area of the *AP vs. Time* curve between given start and end times. The value of an action is the expected result of final evaluation if the action is taken and the policy continues to be followed, which allows actions without an immediate benefit to be scheduled.

VOC dataset and evaluation regime, we are able to obtain better performance than all baselines when there is less time available than is needed to exhaustively run all detectors.

## 2 Recognition Problems and Related Work

Formally, we deal with a dataset of images $\mathcal{D}$, where each image $\mathcal{I}$ contains zero or more objects. Each object is labeled with exactly one category label $k \in \{1, \ldots, K\}$.

The multi-class, multi-label **classification** problem asks whether $\mathcal{I}$ contains at least one object of class $k$. We write the ground truth for an image as $\mathbf{C} = \{C_1, \ldots, C_K\}$, where $C_k \in \{0, 1\}$ is set to 1 if an object of class $k$ is present.

The **detection** problem is to output a list of bounding boxes (sub-images defined by four coordinates), each with a real-valued confidence that it encloses a single instance of an object of class $k$, for each $k$. The answer for a single class $k$ is given by an algorithm $detect(\mathcal{I}, k)$, which outputs a list of sub-image bounding boxes $B$ and their associated confidences.

Performance is evaluated by plotting precision vs. recall across dataset $\mathcal{D}$ (by progressively lowering the confidence threshold for a positive detection). The area under the curve yields the Average Precision (AP) metric, which has become the standard evaluation for recognition performance on challenging datasets in vision [1]. A common measure of a correct detection is the PASCAL overlap: two bounding boxes are considered to match if they have the same class label and the ratio of their intersection to their union is at least $\frac{1}{2}$.

To highlight the hierarchical structure of these problems, we note that the confidences for each sub-image $b \in B$ may be given by $classify(b, k)$, and, more saliently for our setup, correct answer to the detection problem also answers the classification problem.

Multi-class performance is evaluated by averaging the individual per-class AP values. In a specialized system such as the advertising case study from section 1, the metric generalizes to a weighted average, with the weights set by the *values* of the classes.

## 2.1 Related Work

**Object detection** The best recent performance has come from detectors that use gradient-based features to represent objects as either a collection of local patches or as object-sized windows [2, 3]. Classifiers are then used to distinguish between featurizations of a given class and all other possible contents of an image window. Window proposal is most often done exhaustively over the image space, as a "sliding window".

For state-of-the-art performance, the object-sized window models are augmented with parts [4], and the bag-of-visual-words models employ non-linear classifiers [5]. We employ the widely used Deformable Part Model detector [4] in our evaluation.

**Using context** The most common source of context for detection is the *scene* or other non-detector cues; the most common scene-level feature is the GIST [6] of the image. We use this source of scene context in our evaluation.

Inter-object context has also been shown to improve detection [7]. In a standard evaluation setup, inter-object context plays a role only in post-filtering, once all detectors have been run. In contrast, our work leverages inter-object context in the action-planning loop.

A critical summary of the main approaches to using context for object and scene recognition is given in [8]. For the commonly used PASCAL VOC dataset [1], GIST and other sources of context are quantitatively explored in [9].

**Efficiency through cascades** An early success in efficient object detection of a single class uses simple, fast features to build up a *cascade* of classifiers, which then considers image regions in a sliding window regime [10]. Most recently, cyclic optimization has been applied to optimize cascades with respect to feature computation cost as well as classifier performance [11].

Cascades are not dynamic policies: they cannot change the order of execution based on observations obtained during execution, which is our goal.

**Anytime and active classification** This surprisingly little-explored line of work in vision is closest to our approach. A recent application to the problem of visual detection picks features with maximum value of information in a Hough-voting framework [12]. There has also been work on active classification [13] and active sensing [14], in which intermediate results are considered in order to decide on the next classification step. Most commonly, the scheduling in these approaches is greedy with respect to some manual quantity such as expected information gain. In contrast, we learn policies that take actions without any immediate reward.

## 3 Multi-class Recognition Policy

Our goal is a multi-class recognition policy $\pi$ that takes an image $\mathcal{I}$ and outputs a list of multi-class detection results by running detector and global scene *actions* sequentially.

The policy repeatedly selects an action $a_i \in \mathcal{A}$, executes it, receiving observations $o_i$, and then selects the next action. The set of actions $\mathcal{A}$ can include both classifiers and detectors: anything that would be useful for inferring the contents of the image.

Each action $a_i$ has an expected cost $c(a_i)$ of execution. Depending on the setting, the cost can be defined in terms of algorithmic runtime analysis, an idealized property such as number of *flops*, or simply the empirical runtime on specific hardware. We take the empirical approach: every executed action advances $t$, the *time into episode*, by its runtime.

As shown in Figure 1, the system is given two times: the setup time $T_s$ and deadline $T_d$. We want to obtain the best possible answer if stopped at any given time between the setup time and the deadline. A single-number metric that corresponds to this objective is the area captured under the curve between the start and deadline bounds, normalized by the total area. We evaluate policies by this more robust metric and not simply by the final performance at deadline time for the same

reason that Average Precision is used instead of a fixed Precision vs. Recall point in the conventional evaluations.

## 3.1 Sequential Execution

An *open-loop* policy, such as the common classifier cascade [10], takes actions in a sequence that does not depend on observations received from previous actions. In contrast, our goal is to learn a dynamic, or *closed-loop*, policy, which would exploit the signal in scene and inter-object context for a maximally efficient path through the actions.

We refer to the information available to the decision process as the *state s*. The state includes the current estimate of the distribution over class presence variables $P(\mathbf{C}) = \{P(C_0), \ldots, P(C_K)\}$, where we write $P(C_k)$ to mean $P(C_k = 1)$ (class $k$ is present in the image).

Additionally, the state records that an action $a_i$ has been taken by adding it to the initially empty set $\mathcal{O}$ and recording the resulting observations $o_i$. We refer to the current set of observations as $\mathbf{o} = \{o_i | a_i \in \mathcal{O}\}$. The state also keeps track of the time into episode $t$, and the setup and deadline times $T_s, T_d$.

A recognition *episode* takes an image $\mathcal{I}$ and proceeds from the initial state $s^0$ and action $a^0$ to the next pair $(s^1, a^1)$, and so on until $(s^J, a^J)$, where $J$ is the last step of the process with $t \leq T_d$. At that point, the policy is terminated, and a new episode can begin on a new image.

The specific actions we consider in the following exposition are detector actions $a_{det_i}$, where $det_i$ is a detector class $C_i$, and a scene-level context action $a_{gist}$, which updates the probabilities of all classes. Although we avoid this in the exposition, note that our system easily handles multiple detector actions per class.

## 3.2 Selecting actions

As our goal is to pick actions dynamically, we want a function $Q(s, a) : S \times \mathcal{A} \mapsto \mathbb{R}$, where $S$ is the space of all possible states, to assign a value to a potential action $a \in \mathcal{A}$ given the current state $s$ of the decision process. We can then define the policy $\pi$ as simply taking the action with the maximum value:

$$\pi(s) = \operatorname*{argmax}_{a_i \in \mathcal{A} \backslash \mathcal{O}} Q(s, a_i) \tag{1}$$

Although the action space $\mathcal{A}$ is manageable, the space of possible states $S$ is intractable, and we must use function approximation to represent $Q(s, a)$: a common technique in reinforcement learning [15]. We featurize the state-action pair and assume linear structure:

$$Q^\pi(s, a) = \theta_\pi^\top \phi(s, a) \tag{2}$$

The policy's performance at time $t$ is determined by all detections that are part of the set of observations $\mathbf{o}^j$ at the last state $s^j$ before $t$. Recall that detector actions returns lists of detection hypotheses. Therefore, the final AP vs. Time evaluation of an episode is a function $eval(h, T_s, T_d)$ of the history of execution $h = s^0, s^1, \ldots, s^J$. It is precisely the normalized area under the AP vs. Time curve between $T_s$ and $T_d$, as determined by the detections in $\mathbf{o}^j$ for all steps $j$ in the episode.

Note from Figure 3b that this evaluation function is additive per action, as each action $a$ generates observations that may raise or lower the mean AP of the results so far ($\Delta ap$) and takes a certain time ($\Delta t$). We can accordingly represent the final evaluation $eval(h, T_s, T_d)$ in terms of individual action rewards: $\sum_{j=0}^J R(s^j, a^j)$.

Specifically, as shown in Figure 3b, we define the *reward* of an action $a$ as

$$R(s^j, a) = \Delta \mathrm{ap}(t_T^j - \frac{1}{2}\Delta t) \tag{3}$$

where $t_T^j$ is the time left until $T_d$ at state $s^j$, and $\Delta t$ and $\Delta ap$ are the time taken and AP change produced by the action $a$. (We do not account for $T_s$ here for clarity of exposition.)

### 3.3 Learning the policy

The expected value of the final evaluation can be written recursively in terms of the value function:

$$Q^\pi(s^j, a) = \mathbb{E}_{s^{j+1}}[R(s^j, a) + \gamma Q^\pi(s^{j+1}, \pi(s^{j+1}))] \qquad (4)$$

where $\gamma \in [0, 1]$ is the *discount* value.

With $\gamma = 0$, the value function is determined entirely by the immediate reward, and so only completely greedy policies can be learned. With $\gamma = 1$, the value function is determined by the correct expected rewards to the end of the episode. However, a lower value of $\gamma$ mitigates the effects of increasing uncertainty regarding the state transitions over long episodes. We set this meta-parameter of our approach through cross-validation, and find that a mid-level value ($0.4$) works best.

While we can't directly compute the expectation in (4), we can sample it by running actual episodes to gather $< s, a, r, s' >$ samples, where $r$ is the reward obtained by taking action $a$ in state $s$, and $s'$ is the following state.

We then learn the optimal policy by repeatedly gathering samples with the current policy, minimizing the error between the discounted reward to the end of the episode as predicted by our current $Q(s^j, a)$ and the actual values gathered, and updating the policy with the resulting weights.

To ensure sufficient exploration of the state space, we implement $\epsilon$-greedy action selection during training: with a probability that decreases with each training iteration, a random action is selected instead of following the policy. During test time, $\epsilon$ is set to $0.05$.

To prevent overfitting to the training data, we use $L_2$-regularized regression. We run $15$ iterations of accumulating samples by running $350$ episodes, starting with a baseline policy which will be described in section 4, and cross-validating the regularization parameter at each iteration. Samples are not thrown away between iterations.

With pre-computed detections on the PASCAL VOC 2007 dataset, the training procedure takes about $4$ hours on an 8-core *Xeon E5620* machine.

### 3.4 Feature representation

Our policy is at its base determined by a linear function of the features of the state:

$$\pi(s) = \underset{a_i \in \mathcal{A} \setminus \mathcal{O}}{\operatorname{argmax}} \theta_\pi^\top \phi(s, a_i). \qquad (5)$$

We include the following quantities as features $\phi(s, a)$:

| | |
|---|---|
| $P(C_a)$ | The prior probability of the class that corresponds to the detector of action $a$ (omitted for the scene-context action). |
| $P(C_0\|\mathbf{o}) \ldots P(C_K\|\mathbf{o})$ | The probabilities for all classes, conditioned on the current set of observations. |
| $H(C_0\|\mathbf{o}) \ldots H(C_K\|\mathbf{o})$ | The entropies for all classes, conditioned on the current set of observations. |

Additionally, we include the mean and maximum of $[H(C_0|\mathbf{o}) \ldots H(C_K|\mathbf{o})]$, and 4 time features that represent the times until start and deadline, for a total of $F = 1 + 2K + 6$ features.

We note that this setup is commonly used to solve Markov Decision Processes [15]. There are two related limitations of MDPs when it comes to most systems of interesting complexity, however: the state has to be functionally approximated instead of exhaustively enumerated; and some aspects of the state are not observed, making the problem a Partially Observed MDP (POMDP), for which exact solution methods are intractable for all but rather small problems [16]. Our initial solution to the problem of partial observability is to include features corresponding to our level of uncertainty into the feature representation, as in the technique of *augmented* MDPs [17].

To formulate learning the policy as a single regression problem, we represent the features in block form, where $\phi(s, a)$ is a vector of size $F|\mathcal{A}|$, with all values set to $0$ except for the $F$-sized block corresponding to $a$.

As an illustration, we visualize the learned weights on these features in Figure 2, reshaped such that each row shows the weights learned for an action, with the top row representing the scene context action and then next 20 rows corresponding to the PASCAL VOC class detector actions.

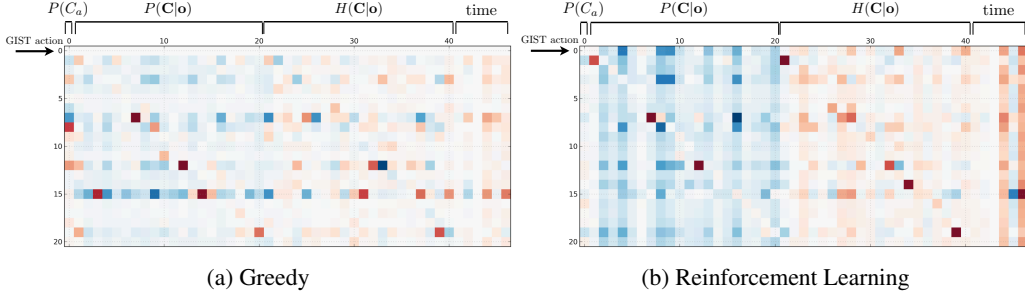

(a) Greedy           (b) Reinforcement Learning

Figure 2: Learned policy weights $\theta_\pi$ (best viewed in color: red corresponds to positive, blue to negative values). The first row corresponds to the scene-level action, which does not generate detections itself but only helps reduce uncertainty about the contents of the image. Note that in the greedy learning case, this action is learned to never be taken, but it is shown to be useful in the reinforcement learning case.

## 3.5 Updating with observations

The bulk of our feature representation is formed by probability of individual class occurrence, conditioned on the observations so far: $P(C_0|\mathbf{o})\ldots P(C_K|\mathbf{o})$. This allows the action-value function to learn correlations between presence of different classes, and so the policy can look for the most probable classes given the observations.

However, higher-order co-occurrences are not well represented in this form. Additionally, updating $P(C_i|\mathbf{o})$ presents choices regarding independence assumptions between the classes. We evaluate two approaches for updating probabilities: *direct* and *MRF*.

In the *direct* method, $P(C_i|\mathbf{o}) = score(C_i)$ if $\mathbf{o}$ includes the observations for class $C_i$ and $P(C_i|\mathbf{o}) = P(C_i)$ otherwise. This means that an observation of class $i$ does not directly influence the estimated probability of any class but $C_i$.

The *MRF* approach employs a pairwise fully-connected Markov Random Field (MRF), as shown in Figure 1, with the observation nodes set to $score(C_i)$ appropriately, or considered unobserved.

The graphical model structure is set as fully-connected, but some classes almost never co-occurr in our dataset. Accordingly, the edge weights are learned with $L_1$ regularization, which obtains a sparse structure [18]. All parameters of the model are trained on fully-observed data, and Loopy Belief Propagation inference is implemented with an open-source graphical model package [19].

An implementation detail: $score(C_i)$ for $a_{det_i}$ is obtained by training a probabilistic classifier on the list of detections, featurized by the top few confidence scores and the total number of detections. Similarly, $score(C_i)$ for $a_{gist}$ is obtained by training probabilistic classifiers on the GIST feature, for all classes.

## 4 Evaluation

We evaluate our system on the multi-class, multi-label detection task, as previously described. We evaluate on a popular detection challenge task: the PASCAL VOC 2007 dataset [1]. This datasets exhibits a rather modest amount of class co-occurrence: the "person" class is highly likely to occur, and less than 10% of the images have more than two classes.

We learn weights on the training and validation sets, and run our policy on all images in the testing set. The final evaluation pools all detections up to a certain time, and computes their multi-class AP per image, averaging over all images. This is done for different times to plot the AP vs. Time curve over the whole dataset. Our method of averaging per-image performance follows [20].

For the detector actions, we use one-vs-all cascaded deformable part-model detectors on a HOG featurization of the image [21], with linear classification of the list of detections as described in the previous section. There are 20 classes in the PASCAL challenge task, so there are 20 detector actions. Running a detector on a PASCAL image takes about 1 second.

We test three different settings of the start and deadline times. In the first one, the start time is immediate and execution is cut off at 20 seconds, which is enough time to run all actions. In the second one, execution is cut off after only 10 seconds. Lastly, we measure performance between 5 seconds and 15 seconds. These operating points show how our method behaves when deployed in different conditions. The results are given in rows of Table 1.

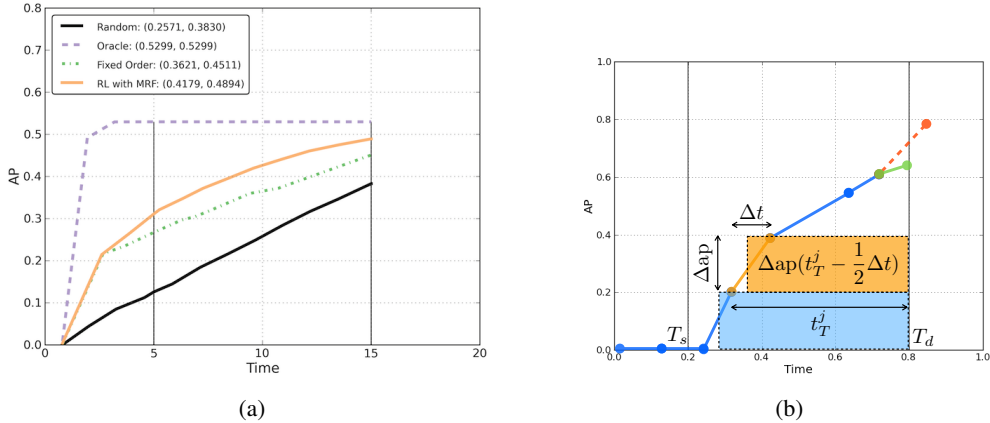

(a)                                                        (b)

Figure 3: (a) AP vs. Time curves for Random, Oracle, the Fixed Order baseline, and our best-performing policy. (b) Graphically representing our reward function, as described in section 3.2.

We establish the first baseline for our system by selecting actions randomly at each step. As shown in Figure 3a, the **Random** policy results in a roughly linear gain of AP vs. time. This is expected: the detectors are capable of obtaining a certain level of performance; if half the detectors are run, the expected performance level is half of the maximum level.

To establish an upper bound on performance, we plot the **Oracle** policy, obtained by re-ordering the actions at the end of each detection episode in the order of AP gains they produced.

We consider another baseline: selecting actions in a fixed order based on the value they bring to the AP vs. Time evaluation, which is roughly proportional to their occurrence probability. We refer to this as **Fixed Order**.

Then there are instantiations of our method, as described in the previous section : **RL w/ Direct** inference and **RL w/ MRF** inference. As the **MRF** model consistently outperformed **Direct** by a small margin, we report results for that model only.

In Figure 3a, we can see that due to the dataset bias, the fixed-order policy performs well at first, as the person class is disproportionately likely to be in the image, but is significantly overtaken by our model as execution goes on and more rare classes have to be detected.

Lastly, we include an additional scene-level GIST feature that updates the posterior probabilities of all classes. This is considered one action, and takes about 0.3 seconds. This setting always uses the MRF model to properly update the class probabilities with GIST observations. This brings another small boost in performance. The results are shown in Table 1.

Visualizing the learned weights in Figure 2, we note that the GIST action is learned to never be taken in the greedy ($\gamma = 0$) setting, but is learned to be taken with a higher value of $\gamma$. It is additionally informative to consider the action trajectories of different policies in Figure 4.

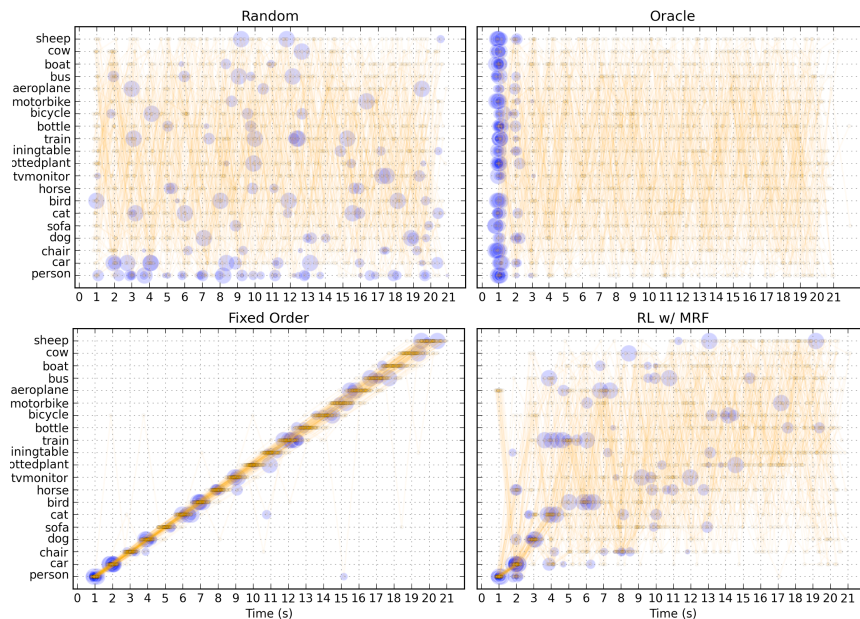

Figure 4: Visualizing the action trajectories of different policies. Action selection traces are plotted in orange over many episodes; the size of the blue circles correspond to the increase in AP obtained by the action. We see that the **Random** policy selects actions and obtains rewards randomly, while the **Oracle** policy obtains all rewards in the first few actions. The **Fixed Order** policy selects actions in a static optimal order. Our policy does not stick a static order but selects actions dynamically to maximize the rewards obtained early on.

Table 1: The areas under the AP vs. Time curve for different experimental conditions.

| Bounds | Random | Fixed Order | RL | RL w/ GIST | Oracle |
|--------|--------|-------------|-------|------------|--------|
| (0,20) | 0.250 | 0.342 | 0.378 | **0.382** | 0.488 |
| (0,10) | 0.119 | 0.240 | 0.266 | **0.267** | 0.464 |
| (5,15) | 0.257 | 0.362 | 0.418 | **0.420** | 0.530 |

## 5 Conclusion

We presented a method for learning "closed-loop" policies for multi-class object recognition, given existing object detectors and classifiers and a metric to optimize. The method learns the optimal policy using reinforcement learning, by observing execution traces in training. If detection on an image is cut off after only half the detectors have been run, our method does $66\%$ better than a random ordering, and $14\%$ better than an intelligent baseline. In particular, our method learns to take action with no intermediate reward in order to improve the overall performance of the system.

As always with reinforcement learning problems, defining the reward function requires some manual work. Here, we derive it for the novel detection AP vs. Time evaluation that we suggest is useful for evaluating efficiency in recognition. Although computation devoted to scheduling actions is less significant than the computation due to running the actions, the next research direction is to explicitly consider this decision-making cost; the same goes for feature computation costs. Additionally, it is interesting to consider actions defined not just by object category but also by spatial region. The code for our method is available[1].

**Acknowledgments**

This research was made with Government support under and awarded by DoD, Air Force Office of Scientific Research, National Defense Science and Engineering Graduate (NDSEG) Fellowship, 32 CFR 168a.

## Footnotes

[1] http://sergeykarayev.com/work/timely/

## References

[1] M Everingham, L Van Gool, C K I Williams, J Winn, and A Zisserman. The PASCAL VOC Challenge. http://www.pascal-network.org/challenges/VOC/, 2010. 2, 3, 6

[2] N Dalal and B Triggs. Histograms of Oriented Gradients for Human Detection. In *CVPR*, pages 886–893, 2005. 3

[3] David G Lowe. Distinctive Image Features from Scale-Invariant Keypoints. *IJCV*, 60(2):91–110, November 2004. 3

[4] Pedro F Felzenszwalb, Ross B Girshick, David McAllester, and Deva Ramanan. Object detection with discriminatively trained part-based models. *PAMI*, 32(9):1627–1645, September 2010. 3

[5] Andrea Vedaldi, Varun Gulshan, Manik Varma, and Andrew Zisserman. Multiple kernels for object detection. *ICCV*, pages 606–613, September 2009. 3

[6] Aude Oliva and Antonio Torralba. Modeling the Shape of the Scene: A Holistic Representation of the Spatial Envelope. *IJCV*, 42(3):145–175, 2001. 3

[7] Antonio Torralba, Kevin P Murphy, and William T Freeman. Contextual Models for Object Detection Using Boosted Random Fields. *MIT CSAIL Technical Report*, 2004. 3

[8] Carolina Galleguillos and Serge Belongie. Context based object categorization: A critical survey. *Computer Vision and Image Understanding*, 114(6):712–722, June 2010. 3

[9] Santosh K Divvala, Derek Hoiem, James H Hays, Alexei A Efros, and Martial Hebert. An empirical study of context in object detection. In *CVPR*, pages 1271–1278, June 2009. 3

[10] Paul Viola and Michael Jones. Rapid object detection using a boosted cascade of simple features. In *CVPR*, 2001. 3, 4

[11] Minmin Chen, Zhixiang (Eddie) Xu, Kilian Q Weinberger, Olivier Chapelle, and Dor Kedem. Classifier Cascade for Minimizing Feature Evaluation Cost. In *AISTATS*, 2012. 3

[12] Sudheendra Vijayanarasimhan and Ashish Kapoor. Visual Recognition and Detection Under Bounded Computational Resources. In *CVPR*, pages 1006–1013, 2010. 3

[13] Tianshi Gao and Daphne Koller. Active Classification based on Value of Classifier. In *NIPS*, 2011. 3

[14] Shipeng Yu, Balaji Krishnapuram, Romer Rosales, and R Bharat Rao. Active Sensing. In *AISTATS*, pages 639–646, 2009. 3

[15] Richard S Sutton and Andrew G Barto. *Reinforcement Learning: An Introduction*. MIT Press, 1998. 4, 5

[16] Nicholas Roy and Geoffrey Gordon. Exponential Family PCA for Belief Compression in POMDPs. In *NIPS*, 2002. 5

[17] Cody Kwok and Dieter Fox. Reinforcement Learning for Sensing Strategies. In *IROS*, 2004. 5

[18] Su-In Lee, Varun Ganapathi, and Daphne Koller. Efficient Structure Learning of Markov Networks using L1-Regularization. In *NIPS*, 2006. 6

[19] Ariel Jaimovich and Ian Mcgraw. FastInf: An Efficient Approximate Inference Library. *Journal of Machine Learning Research*, 11:1733–1736, 2010. 6

[20] Chaitanya Desai, Deva Ramanan, and Charless Fowlkes. Discriminative models for multi-class object layout. In *ICCV*, pages 229–236, September 2009. 6

[21] Pedro F Felzenszwalb, Ross B Girshick, and David McAllester. Cascade object detection with deformable part models. In *CVPR*, pages 2241–2248. IEEE, June 2010. 7

